# Reinforcement Learning by Probability Matching

Philip N. Sabes
sabes@psyche.mit.edu

Michael I. Jordan
jordan@psyche.mit.edu

Department of Brain and Cognitive Sciences
Massachusetts Institute of Technology
Cambridge, MA 02139

## Abstract

We present a new algorithm for associative reinforcement learning. The algorithm is based upon the idea of matching a network's output probability with a probability distribution derived from the environment's reward signal. This Probability Matching algorithm is shown to perform faster and be less susceptible to local minima than previously existing algorithms. We use Probability Matching to train mixture of experts networks, an architecture for which other reinforcement learning rules fail to converge reliably on even simple problems. This architecture is particularly well suited for our algorithm as it can compute arbitrarily complex functions yet calculation of the output probability is simple.

## 1 INTRODUCTION

The problem of learning associative networks from scalar reinforcement signals is notoriously difficult. Although general purpose algorithms such as REINFORCE (Williams, 1992) and Generalized Learning Automata (Phansalkar, 1991) exist, they are generally slow and have trouble with local minima. As an example, when we attempted to apply these algorithms to mixture of experts networks (Jacobs et al., 1991), the algorithms typically converged to the local minimum which places the entire burden of the task on one expert.

Here we present a new reinforcement learning algorithm which has faster and more reliable convergence properties than previous algorithms. The next section describes the algorithm and draws comparisons between it and existing algorithms. The following section details its application to Gaussian units and mixtures of Gaussian experts. Finally, we present empirical results.

## 2   REINFORCEMENT PROBABILITY MATCHING

We begin by formalizing the learning problem. Given an input $\mathbf{x} \in \mathcal{X}$ from the environment, the network must select an output $\mathbf{y} \in \mathcal{Y}$. The network then receives a scalar reward signal $r$, with a mean $\bar{r}$ and distribution that depend on $\mathbf{x}$ and $\mathbf{y}$. The goal of the learner is to choose an output which maximizes the expected reward. Due to the lack of an explicit error signal, the learner must choose its output stochastically, exploring for better rewards. Typically the learner starts with a parameterized form for the conditional output density $p_\theta(\mathbf{y}|\mathbf{x})$, and the learning problem becomes one of finding the parameters $\theta$ which maximize the expected reward:

$$J_r(\theta) = \int_{\mathcal{X},\mathcal{Y}} p(\mathbf{x})p_\theta(\mathbf{y}|\mathbf{x})\bar{r}(\mathbf{x},\mathbf{y})d\mathbf{y}d\mathbf{x}.$$

We present an alternative route to the maximum expected reward cost function, and in doing so derive a novel learning rule for updating the network's parameters. The learner's task is to choose from a set of conditional output distributions based on the reward it receives from the environment. These rewards can be thought of as inverse energies; input/output pairs that receive high rewards are low energy and are preferred by the environment. Energies can always be converted into probabilities through the Boltzmann distribution, and so we can define the environment's conditional distribution on $\mathcal{Y}$ given $\mathbf{x}$,

$$p^*(\mathbf{y}|\mathbf{x}) = \frac{\exp(-T^{-1}E(\mathbf{x},\mathbf{y}))}{Z_T(\mathbf{x})} = \frac{\exp(T^{-1}\bar{r}(\mathbf{x},\mathbf{y}))}{Z_T(\mathbf{x})},$$

where $T$ is a temperature parameter and $Z_T(\mathbf{x})$ is a normalization constant which depends on $T$. This distribution can be thought of as representing the environment's ideal input-output mapping, high reward input-output pairs being more typical or likely than low reward pairs. The temperature parameter determines the strength of this preference: when $T$ is infinity all outputs are equally likely; when $T$ is zero only the highest reward output is chosen. This new distribution is a purely theoretical construct, but it can be used as a target distribution for the learner. If the $\theta$ are adjusted so that $p_\theta(\mathbf{y}|\mathbf{x})$ is nearly equal to $p^*(\mathbf{y}|\mathbf{x})$, then the network's output will typically result in high rewards.

The agreement between the network and environment conditional output densities can be measured with the Kullback-Liebler (KL) divergence:

$$KL(p \parallel p^*) = -\int_{\mathcal{X},\mathcal{Y}} p(\mathbf{x})p_\theta(\mathbf{y}|\mathbf{x})\left[\log p^*(\mathbf{y}|\mathbf{x}) - \log p_\theta(\mathbf{y}|\mathbf{x})\right]d\mathbf{y}d\mathbf{x} \qquad (1)$$

$$= -\frac{1}{T}\int_{\mathcal{X},\mathcal{Y}} p(\mathbf{x})p_\theta(\mathbf{y}|\mathbf{x})\left[\bar{r}(\mathbf{x},\mathbf{y}) - T\hat{r}_\theta(\mathbf{x},\mathbf{y})\right]d\mathbf{y}d\mathbf{x} + \int_{\mathcal{X}} p(\mathbf{x})\log Z_T(\mathbf{x})d\mathbf{x},$$

where $\hat{r}_\theta(\mathbf{x},\mathbf{y})$ is defined as the logarithm of the conditional output probability and can be thought of as the network's estimate of the mean reward. This cost function is always greater than or equal to zero, with equality only when the two probability distributions are identical.

Keeping only the part of Equation 1 which depends on $\theta$, we define the Probability Matching (PM) cost function:

$$J_{PM}(\theta) = -\int_{\mathcal{X},\mathcal{Y}} p(\mathbf{x})p_\theta(\mathbf{y}|\mathbf{x})\left[\bar{r}(\mathbf{x},\mathbf{y}) - T\hat{r}_\theta(\mathbf{x},\mathbf{y})\right]d\mathbf{y}d\mathbf{x} = -J_r(\theta) - TS(p_\theta)$$

The PM cost function is analogous to a free energy, balancing the energy, in the form of the negative of the average reward, and the entropy $S(p_\theta)$ of the output

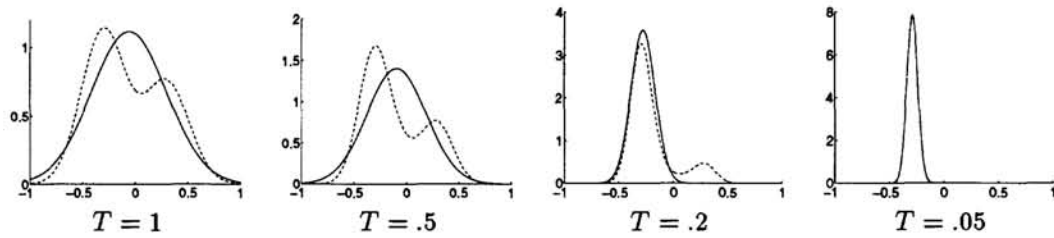

$$T = 1 \qquad\qquad T = .5 \qquad\qquad T = .2 \qquad\qquad T = .05$$

Figure 1: $p^*$'s (dashed) and PM optimal Gaussians (solid) for the same bimodal reward function and various temperatures. Note the differences in scale.

distribution. A higher $T$ corresponds to a smoother target distribution and tilts the balance of the cost function in favor of the entropy term, making diffuse output distributions more favorable. Likewise, a small $T$ results in a sharp target distribution placing most of the weight on the reward dependent term of cost function, which is always optimized by the singular solution of a spike at the highest reward output.

Although minimizing the PM cost function will result in sampling most often at high reward outputs, it will not optimize the overall expected reward if $T > 0$. There are two reasons for this. First, the output $\mathbf{y}$ which maximizes $\hat{r}_\theta(\mathbf{x}, \mathbf{y})$ may not maximize $\bar{r}(\mathbf{x}, \mathbf{y})$. Such an example is seen in the first panel of Figure 1: the network's conditional output density is a Gaussian with adjustable mean and variance, and the environment has a bimodal reward function and $T = 1$. Even in the realizable case, however, the network will choose outputs which are suboptimal with respect to its own predicted reward, with the probability of choosing output $\mathbf{y}$ falling off exponentially with $\hat{r}_\theta(\mathbf{x}, \mathbf{y})$. The key point here is that early in learning this non-optimality is exactly what is desired. The PM cost function forces the learner to maintain output density everywhere the reward, as measure by $p^{*1/T}$, is not much smaller than its maximum. When $T$ is high, the rewards are effectively flattened and even fairly small rewards look big. This means that a high temperature ensures that the learner will explore the output space.

Once the network is nearly PM optimal, it would be advantageous to "sharpen up" the conditional output distribution, sampling more often at outputs with higher predicted rewards. This translates to decreasing the entropy of the output distribution or lowering $T$. Figure 1 shows how the PM optimal Gaussian changes as the temperature is lowered in the example discussed above; at very low temperatures the output is almost always near the mode of the target distribution. In the limit of $T = 0$, $J_{PM}$ becomes original reward maximization criterion $J_r$. The idea of the Probability Matching algorithm is to begin training with a large $T$, say unity, and gradually decrease it as the performance improves, effectively shifting the bias of the learner from exploration to exploitation.

We now must find an update rule for $\theta$ which minimizes $J_{PM}(\theta)$. We proceed by looking for a stochastic gradient descent step. Differentiating the cost function gives

$$\nabla_\theta J_{PM}(\theta) = -\int_{\mathcal{X}, \mathcal{Y}} p(\mathbf{x}) p_\theta(\mathbf{y}|\mathbf{x}) \left[\bar{r}(\mathbf{x}, \mathbf{y}) - T\hat{r}_\theta(\mathbf{x}, \mathbf{y})\right] \nabla_\theta \hat{r}_\theta(\mathbf{x}, \mathbf{y}) dy dx.$$

Thus, if after every action the parameters are updated by the step

$$\triangle\theta = \alpha \left[r - T\hat{r}_\theta(\mathbf{x}, \mathbf{y})\right] \nabla_\theta \hat{r}_\theta(\mathbf{x}, \mathbf{y}), \qquad\qquad (2)$$

where alpha is a constant which can vary over time, then the parameters will on average move down the gradient of the PM cost function. Note that any quantity

which does not depend on $\mathbf{y}$ or $r$ can be added to the difference in the update rule, and the expected step will still point along the direction of the gradient.

The form of Equation 2 is similar to the REINFORCE algorithm (Williams, 1992), whose update rule is

$$\triangle\theta = \alpha(r - b)\nabla_\theta \log p_\theta(\mathbf{y}|\mathbf{x}),$$

where $b$, the reinforcement baseline, is a quantity which does not depend on $\mathbf{y}$ or $r$. Note that these two update rules are identical when $T$ is zero.[1] The advantage of the PM rule is that it allows for an early training phase which encourages exploration without forcing the output distribution to converge on suboptimal outputs. This will lead to striking qualitative differences in the performance of the algorithm for training mixtures of Gaussian experts.

## 3  UPDATE RULES FOR GAUSSIAN UNITS AND MIXTURES OF GAUSSIAN EXPERTS

We employ Gaussian units with mean $\mu = \mathbf{w}^{\mathrm{T}}\mathbf{x}$ and covariance $\sigma^2\mathbf{I}$. The learner must select the matrix $\mathbf{w}$ and scalar $\sigma$ which minimize $J_{PM}(\mathbf{w}, \sigma)$. Applying the update rule in Equation 2, we get

$$\begin{aligned}
\triangle\mathbf{w} &= \alpha\left[r - T\hat{r}(\mathbf{x}, \mathbf{y})\right]\frac{1}{\sigma^2}(\mathbf{y} - \mu)^{\mathrm{T}}\mathbf{x} \\
\triangle\sigma &= \alpha\left[r - T\hat{r}(\mathbf{x}, \mathbf{y})\right]\frac{1}{\sigma^2}\left(\frac{\|\mathbf{y} - \mu\|^2}{\sigma^2} - 1\right).
\end{aligned}$$

In practice, for both single Gaussian units and the mixtures presented below we avoid the issue of constraining $\sigma > 0$ by updating $\log \sigma$ directly.

We can generalize the linear model by considering a conditional output distribution in the form of a mixture of Gaussian experts (Jacobs et al., 1991),

$$p(\mathbf{y}|\mathbf{x}) = \sum_{i=1}^{N} g_i(\mathbf{x})(2\pi\sigma_i^2)^{-\frac{1}{2}}\exp(-\frac{1}{2\sigma_i^2}\|\mathbf{y} - \mu_i\|^2).$$

Expert $i$ has mean $\mu_i = \mathbf{w}_i^{\mathrm{T}}\mathbf{x}$ and covariance $\sigma_i^2\mathbf{I}$. The prior probability given $\mathbf{x}$ of choosing expert $i$, $g_i(\mathbf{x})$, is determined by a single layer gating network with weight matrix $\mathbf{v}$ and softmax output units. The gating network learns a soft partitioning of the input space into regions for which each expert is responsible.

Again, we can apply Equation 2 to get the PM update rules:

$$\begin{aligned}
\triangle\mathbf{v}_i &= \alpha\left[r - T\hat{r}(\mathbf{x}, \mathbf{y})\right](h_i - g_i)\mathbf{x} \\
\triangle\mathbf{w}_i &= \alpha\left[r - T\hat{r}(\mathbf{x}, \mathbf{y})\right]h_i\frac{1}{\sigma_i^2}(\mathbf{y} - \mu_i)^{\mathrm{T}}\mathbf{x} \\
\triangle\sigma_i &= \alpha\left[r - T\hat{r}(\mathbf{x}, \mathbf{y})\right]h_i\frac{1}{\sigma_i^2}\left(\frac{\|\mathbf{y} - \mu_i\|^2}{\sigma_i^2} - 1\right),
\end{aligned}$$

where $h_i = g_i p_i(\mathbf{y}|\mathbf{x})/p(\mathbf{y}|\mathbf{x})$ is the posterior probability of choosing expert $i$ given $\mathbf{y}$. We note that the PM update rules are equivalent to the supervised learning gradient descent update rules in (Jacobs et al., 1991) modulated by the difference between the actual and expected rewards.

Table 1: Convergence times and gate entropies for the linear example (standard errors in parentheses). Convergence times: An experiment consisting of 50 runs was conducted for each algorithm, with a wide range of learning rates and both reward functions. Best results for each algorithm are reported. Entropy: Values are averages over the last 5,000 time steps of each run. 20 runs of 50,000 time steps were conducted.

| Algorithm | Convergence Time | Entropy |
|---|---|---|
| PM, $T = 1$ | 1088 (43) | .993 (.001) |
| PM, $T = .5$ | — | .97 (.02) |
| PM, $T = .1$ | — | .48 (.04) |
| REINFORCE | 2998 (183) | .21 (.03) |
| REINF-COMP | 1622 (46) | .21 (.03) |

Both the $h_i$ and $\hat{r}$ depend on the overall conditional probability $p(\mathbf{y}|\mathbf{x})$, which in turn depends on each $p_i(\mathbf{y}|\mathbf{x})$. This adds an extra step to the training procedure. After receiving the input $\mathbf{x}$, the network chooses an expert based on the priors $g_i(\mathbf{x})$ and an output $\mathbf{y}$ from the selected expert's output distribution. The output is then sent back to each of the experts in order to compute the likelihood of their having generated it. Given the set of $p_i$'s, the network can update its parameters as above.

## 4  SIMULATIONS

We present three examples designed to explore the behavior of the Probability Matching algorithm. In each case, networks were trained using Probability Matching, REINFORCE, and REINFORCE with reinforcement comparison (REINF-COMP), where a running average of the reward is used as a reinforcement baseline (Sutton, 1984). In the first two examples an optimal output function $\mathbf{y}^*(\mathbf{x})$ was chosen and used to calculate a noisy error, $\varepsilon = \|\mathbf{y} - \mathbf{y}^*(\mathbf{x}) - \mathbf{z}\|$, where $\mathbf{z}$ was i.i.d. zero-mean Gaussian with $\sigma = .1$. The error signal determined the reward by one of two functions, $r = -\varepsilon^2/2$ or $\exp(-\varepsilon^2/2)$. When the RMSE between the network mean and the optimal output was less that .05 the network was said to have converged.

### 4.1  A Linear Example

In this example $\mathbf{x}$ was chosen uniformly from $[-1, 1]^2 \times \{1\}$, and the optimal output was $\mathbf{y}^* = A\mathbf{x}$, for a $2 \times 3$ matrix $A$. A mixture of three Gaussian experts was trained. The details of the simulation and results for each algorithm are shown in Table 1. Probability Matching with constant $T = 1$ shows almost a threefold reduction in training time compared to REINFORCE and about a 50% improvement over REINF-COMP.

The important point of this example is the manner in which the extra Gaussian units were employed. We calculated the entropy of the gating network, normalized so that a value of one means that each expert has equal probability of being chosen and a value of zero means that only one expert is ever chosen. The values after 50,000 time steps are shown in the second column of Table 1. When $T \approx 1$, the Probability Matching algorithm gives the three experts roughly equal priors. This is due to the fact that small differences in the experts' parameters lead to increased output entropy if all experts are used. REINFORCE on the other hand always converges to a solution which employs only one expert. This difference in the behavior of the algorithms will have a large qualitative effect in the next example.

Table 2: Results for absolute value. The percentage of trials that converged and the average time to convergence for those trials. Standard errors are in parentheses. 50 trials were conducted for a range of learning rates and with both reward functions; the best results for each algorithm are shown.

| Algorithm | Successful Trials | Convergence Time |
|---|---|---|
| PM | 100% | 6,052 (313) |
| REINFORCE | 48% | 76,775 (3,329) |
| REINF-COMP | 38% | 42,105 (3,869) |

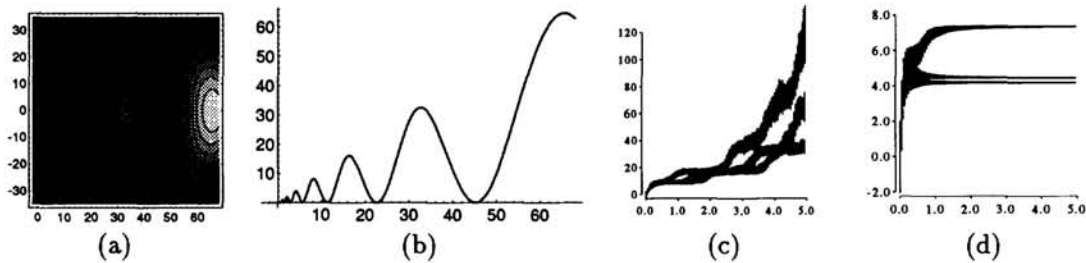

| (a) | (b) | (c) | (d) |

Figure 2: Example 4.3. The environment's probability distribution for $T = 1$: (a) density plot of $p^*$ vs. $\mathbf{y}/x$, (b) cross-sectional view with $y_2 = 0$. Locally weighted mean and variance of $y_2/x$ over representative runs: (c) $T = 1$, (d) $T = 0$ (i.e. REINFORCE).

## 4.2 Absolute Value

We used a mixture of two Gaussian units to learn the absolute value function. The details of the simulation and the best results for each algorithm are shown in Table 2. Probability Matching with constant $T = 1$ converged to criterion on every trial, in marked contrast to the REINFORCE algorithm. With no reinforcement baseline, REINFORCE converged to criterion in only about half of the cases, less with reinforcement comparison. In almost all of the trials that didn't converge, only one expert was active on the domain of the input. Neither version of REINFORCE ever converged to criterion in less than 14,000 time steps.

This example highlights the advantage of the Probability Matching algorithm. During training, all three algorithms initially use both experts to capture the overall mean of the data. REINFORCE converges on this local minimum, cutting one expert off before it has a chance to explore the rest of the parameter space. The Probability Matching algorithm keeps both experts in use. Here, the more conservative approach leads to a stark improvement in performance.

## 4.3 An Example with Many Local Maxima

In this example, the learner's conditional output distribution was a bivariate Gaussian with $\boldsymbol{\mu} = [w_1, w_2]^T x$, and the environment's rewards were a function of $\mathbf{y}/x$. The optimal output distribution $p^*(\mathbf{y}/x)$ is shown in Figures 2(a,b). These figures can also be interpreted as the expected value of $p^*$ for a given $\mathbf{w}$. The weight vector is initially chosen from a uniform distribution over $[-.2, .2]^2$, depicted as the very small while dot in Figure 2(a). There are a series of larger and larger local maxima off to the right, with a peak of height $2^n$ at $w_1 = 2^n$.

The results are shown in Table 3. REINFORCE, both with and without reinforcement comparison, never got past third peak; the variance of the Gaussian unit would

Table 3: Results for Example 4.3. These values represent 20 runs for 50,000 time steps each. The first and second columns correspond to number of the peak the learner reached.

| Algorithm | Mean Final $\log_2 w_1$ | Range of Final $\log_2 w_1$'s | Mean Final $\sigma$ |
|---|---|---|---|
| PM, $T = 2$ | 28,8 | [19.1,51.0] | $> 10^5$ |
| PM, $T = 1$ | 6.34 | [5.09,8.08] | 13.1 |
| PM, $T = .5$ | 3.06 | [3.04,3.07] | .40 |
| REINFORCE | 2.17 | [2.00,2.90] | .019 |
| REINF-COMP | 2.05 | [2.05,2.06] | .18 |

very quickly close down to a small value making further exploration of the output space impossible. Probability Matching, on the other hand, was able to find greater and greater maxima, with the variance growing adaptively to match the local scale of the reward function. These differences can be clearly seen in Figures 2(c,d), which show typical behavior for the Probability Matching algorithm with $T = 1$ and $T = 0$.

## 5    CONCLUSION

We have presented a new reinforcement learning algorithm for associative networks which converges faster and more reliably than existing algorithms. The strength of the Probability Matching algorithm is that it allows for a better balance between exploration of the output space and and exploitation of good outputs. The parameter $T$ can be adjusted during learning to allow broader output distributions early in training and then to force the network to sharpen up its distribution once nearly optimal parameters have been found.

Although the applications in this paper were restricted to networks with Gaussian units, the Probability Matching algorithm can be applied to any reinforcement learning task and any conditional output distribution. It could easily be employed, for example, on classification problems using logistic or multinomial (softmax) output units or mixtures of such units. Finally, the simulations presented in this paper are of simple examples. Preliminary results indicate that the advantages of the Probability Matching algorithm scale up to larger, more interesting problems.

## Footnotes

[1]This fact implies that the REINFORCE step is in the direction of the gradient of $J_R(\theta)$, as shown by (Williams, 1992). See Williams and Peng, 1991, for a similar REINFORCE plus entropy update rule.

## References

Jacobs, R. A., Jordan, M. I., Nowlan, S. J., and Hinton, G. E. (1991). Adaptive mixtures of local experts. *Neural Computation*, 3:79–87.

Phansalkar, V. V. (1991). *Learning automata algorithms for connectionist systems – local and global convergence*. PhD Thesis, Dept. of Electrical Engineering, India Institute of Science, Bangalore.

Sutton, R. S. (1984). *Temporal credit assignment in reinforcement learning*. PhD Thesis, Dept. of Computer and Information Science, University of Massachusetts, Amherst, MA.

Williams, R. J. (1992). Simple statistical gradient-following algorithms for connectionist reinforcement learning. *Machine Learning*, 8:229–256.

Williams, R. J. and Peng, J. (1991). Function optimization using connectionist reinforcement learning algorithms. *Connection Science*, 3:241–268.
